# Salient Contour Extraction by Temporal Binding in a Cortically-Based Network

**Shih-Cheng Yen** and **Leif H. Finkel**
Department of Bioengineering and
Institute of Neurological Sciences
University of Pennsylvania
Philadelphia, PA 19104, U. S. A.
syen@jupiter.seas.upenn.edu
leif@jupiter.seas.upenn.edu

## Abstract

It has been suggested that long-range intrinsic connections in striate cortex may play a role in contour extraction (Gilbert *et al.*, 1996). A number of recent physiological and psychophysical studies have examined the possible role of long range connections in the modulation of contrast detection thresholds (Polat and Sagi, 1993,1994; Kapadia *et al.*, 1995; Kovács and Julesz, 1994) and various pre-attentive detection tasks (Kovács and Julesz, 1993; Field *et al.*, 1993). We have developed a network architecture based on the anatomical connectivity of striate cortex, as well as the temporal dynamics of neuronal processing, that is able to reproduce the observed experimental results. The network has been tested on real images and has applications in terms of identifying salient contours in automatic image processing systems.

## 1    INTRODUCTION

Vision is an active process, and one of the earliest, preattentive actions in visual processing is the identification of the salient contours in a scene. We propose that this process depends upon two properties of striate cortex: the pattern of horizontal connections between orientation columns, and temporal synchronization of cell responses. In particular, we propose that perceptual salience is directly related to the degree of cell synchronization.

We present results of network simulations that account for recent physiological and psychophysical "pop-out" experiments, and which successfully extract salient contours from real images.

## 2    MODEL ARCHITECTURE

Linear quadrature steerable filter pyramids (Freeman and Adelson, 1991) are used to model the response characteristics of cells in primary visual cortex. Steerable filters are computationally efficient as they allow the energy at any orientation and spatial frequency to be calculated from the responses of a set of basis filters. The fourth derivative of a Gaussian and its Hilbert transform were used as the filter kernels to approximate the shape of the receptive fields of simple cells.

The model cells are interconnected by long-range horizontal connections in a pattern similar to the co-circular connectivity pattern of Parent and Zucker (1989), as well as the "association field" proposed by Field *et al.* (1993). For each cell with preferred orientation, $\theta$, the orientations , $\phi$, of the pre-synaptic cell at position $(i,j)$ relative to the post-synaptic cell, are specified by:

$$\phi(\theta, i, j) = 2 \tan^{-1}\left(\frac{j}{i}\right) - \theta$$

(see Figure 1a). These excitatory connections are confined to two regions, one flaring out along the axis of orientation of the cell (co-axial), and another confined to a narrow zone extending orthogonally to the axis of orientation (trans-axial). The fan-out of the co-axial connections is limited to low curvature deviations from the orientation axis while the trans-axial connections are limited to a narrow region orthogonal to the cell's orientation axis. These constraints are expressed as:

$$\Gamma(\theta, i, j, \psi) = \begin{cases} 1, & \text{if } \tan^{-1}\left(\frac{j}{i}\right) - \theta < \kappa, \\ 1, & \text{if } \tan^{-1}\left(\frac{j}{i}\right) - \theta = \frac{\pi}{2} \pm \varepsilon, \\ 0, & \text{otherwise.} \end{cases}$$

where $\kappa$ represents the maximum angular deviation from the orientation axis of the post-synaptic cell and $\varepsilon$ represents the maximum angular deviation from the orthogonal axis of the post-synaptic cell. Connection weights decrease for positions with increasing angular deviation from the orientation axis of the cell, as well as positions with increasing distance, in agreement with the physiological and psychophysical findings. Figure 1b illustrates the connectivity pattern. There is physiological, anatomical and psychophysical evidence consistent with the existence of both sets of connections (Nelson and Frost, 1985; Kapadia *et al.*, 1995; Rockland and Lund, 1983; Lund *et al.*, 1985; Fitzpatrick, 1996; Polat and Sagi, 1993, 1994).

Cells that are facilitated by the connections inhibit neighboring cells that lie outside the facilitatory zones. The magnitude of the inhibition is such that only cells receiving strong support are able to remain active. This is consistent with the physiological findings of Nelson and Frost (1985) and Kapadia *et al.* (1995) as well as the intra-cellular recordings of Weliky et al. (1995) which show EPSPs followed by IPSPs when the long-distance connections were stimulated. This inhibition is thought to occur through di-synaptic pathways.

In the model, cells are assumed to enter a "bursting mode" in which they synchronize with other bursting cells. In cortex, bursting has been associated with supragranular "chattering cells" (Gray and McCormick (1996). In the model, cells that enter the bursting

mode are modeled as homogeneous coupled neural oscillators with a common fundamental frequency but different phases (Kopell and Ermentrout, 1986; Baldi and Meir, 1990). The phase of each oscillator is modulated by the phase of the oscillators to which it is coupled. Oscillators are coupled only to other oscillators with which they have strong, reciprocal, connections. The oscillators synchronize using a simple phase averaging rule:

$$\Theta_i(t) = \frac{\sum w_{ij} \Theta_j(t-1)}{\sum w_{ij}}, \qquad w_{ii} = 1$$

where $\Theta$ represents the phase of the oscillator and $w_{ij}$ represents the weight of the connection between oscillator $i$ and $j$. The oscillators synchronize iteratively with synchronization defined as the following condition:

$$\left| \Theta_i(t) - \Theta_j(t) \right| < \delta, \quad i, j \in C, \quad t < t_{max}$$

where $C$ represents all the coupled oscillators on the same contour, $\delta$ represents the maximum phase difference between oscillators, and $t_{max}$ represents the maximum number of time steps the oscillators are allowed to synchronize. The salience of the chain is then represented by the sum of the activities of all the synchronized elements in the group, $C$. The chain with the highest salience is chosen as the output of the network. This allows us to compare the output of the model to psychophysical results on contour extraction.

It has been postulated that the 40 Hz oscillations observed in the cortex may be responsible for perceptual binding across different cortical regions (Singer and Gray, 1995). Recent studies have questioned the functional significance and even the existence of these oscillations (Ghose and Freeman, 1992; Bair *et al.*, 1994). We use neural oscillators only as a simple functional means of computing synchronization and make no assumption regarding their possible role in cortex.

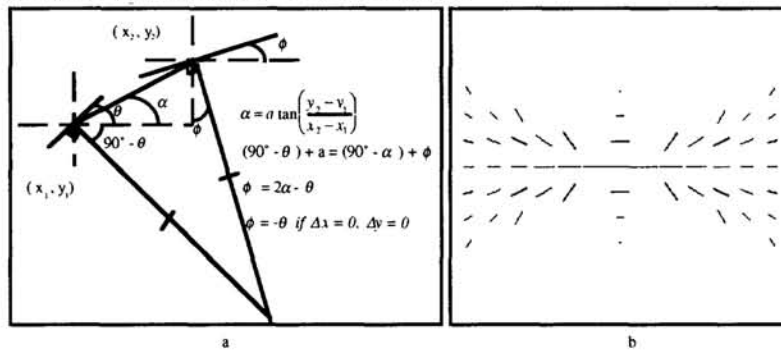

Figure 1: a) Co-circularity constraint. b) Connectivity pattern of a horizontally oriented cell. Length of line indicates connection strength.

# 3 RESULTS

This model was tested by simulating a number of psychophysical experiments. A number of model parameters remain to be defined through further physiological and psychophysical experiments, thus we only attempt a qualitative fit to the data. All simulations were conducted with the same parameter set.

## 3.1 EXTRACTION OF SALIENT CONTOURS

Using the same methods as Field *et al.* (1993), we tested the model's ability to extract contours embedded in noise (see Figure 2). Pairs of stimulus arrays were presented to the

network, one array contains a contour, the other contains only randomly oriented elements. The network determines the stimulus containing the contour with the highest salience. Network performance was measured by computing the percentage of correct detection. The network was tested on a range of stimulus variables governing the target contour: 1) the angle, $\beta$, between elements on a contour, 2) the angle between elements on a contour but with the elements aligned orthogonal to the contour passing through them, 3) the angle between elements with a random offset angle, $\pm\alpha$, with respect to the contour passing through them, and 4) average separation of the elements. 500 simulations were run at each data point. The results are shown in Figure 2. The model shows good qualitative agreement with the psychophysical data. When the elements are aligned, the performance of the network is mostly modulated by the co-axial connections, whereas when the elements are oriented orthogonal to the contour, the trans-axial connections mediate performance. Both the model and human subjects are adversely affected as the weights between consecutive elements decrease in strength. This reduces the length of the contour and thus the saliency of the stimulus.

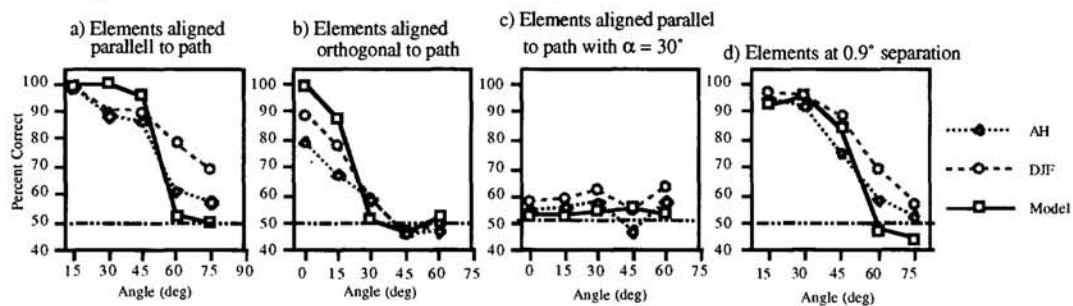

Figure 2: Simulation results are compared to the data from 2 subjects (AH, DJF) in Field *et al.* (1993). Stimuli consisted of 256 randomly oriented Gabor patches with 12 elements aligned to form a contour. Each data point represents results for 500 simulations.

## 3.2   EFFECTS OF CONTOUR CLOSURE

In a series of experiments using similar stimuli to Field *et al.* (1993), Kovács and Julesz (1993) found that closed contours are much more salient than open contours. They reported that when the inter-element spacing between all elements was gradually increased, the maximum inter-element separation for detecting closed contours, $\Delta_c$ (defined at 75% performance), is higher than that for open contours, $\Delta_o$. In addition, they showed that when elements spaced at $\Delta_o$ are added to a "jagged" (open) contour, the saliency of the contour increases monotonically but when elements spaced at $\Delta_c$ are added to a circular contour, the saliency does not change until the last element is added and the contour becomes closed. In fact, at $\Delta_c$, the contour is not salient until it is closed, at which point it suddenly "pops-out" (see Figure 3c). This finding places a strong constraint on the computation of saliency in visual perception.

Interestingly, it has been shown that synchronization in a chain of coupled neural oscillators is enhanced when the chain is closed (Kopell and Ermentrout, 1986; Ermentrout, 1985; Somers and Kopell, 1993). This property seems to be related to the differences in boundary effects on synchronization between open and closed chains and appears to hold across different families of coupled oscillators. It has also been shown that synchronization is dependent on the coupling between oscillators -- the stronger the coupling, the better the synchronization, both in terms of speed and coherence (Somers

and Kopell, 1993; Wang, 1995). We believe these findings may apply to the psychophysical results.

As in Kovács and Julesz (1993), the network is presented with two stimuli, one containing a contour and the other made up of all randomly oriented elements. The network picks the stimulus containing the synchronized contour with the higher salience. In separate trials, the threshold for maximum separation between elements was determined for open and closed contours. The ratio of the separation of the background elements to the that of elements on a closed curve, $\varphi_c$, was found to be 0.6 (which is similar to the threshold of 0.65 recently reported by Kovács *et al.*, 1996), whereas the ratio for open contours, $\varphi_o$, was found to be 0.9. ($\Delta$ is the threshold separation of contour elements, $\varphi$, at a particular background separation). We then examined the changes in salience for open and closed contours. The performance of the network was measured as additional elements were added to an initial short contour of elements. The results are shown in Figure 3b. At $\varphi_o$, both open and closed contours are synchronized but at $\varphi_c$, elements are synchronized only when the chains are closed. If salience can only be computed for synchronized contours, then as additional elements are added to an open chain at $\varphi_o$, the salience would increase since the whole chain is synchronized. On the other hand, at $\varphi_c$, as long as the last element is missing, the chain is really an open chain, and since $\varphi_c$ is smaller than $\varphi_o$, the elements on the chain will not be able to synchronize and adding elements has no effect on salience. Once the last element is added, the chain is immediately able to synchronize and the salience of the contour increases dramatically and causes the contour to "pop-out".

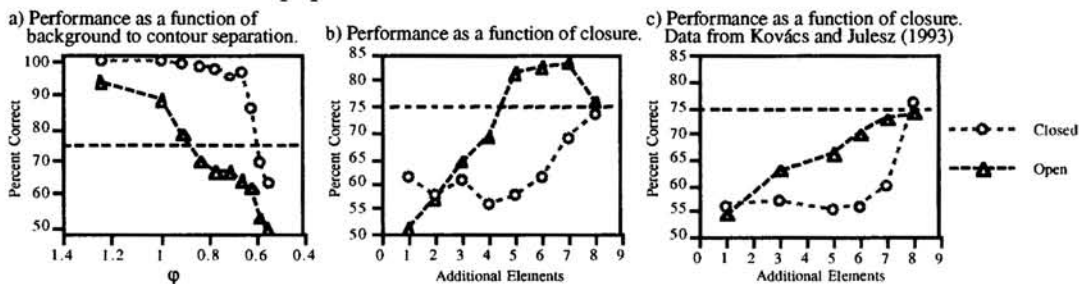

Figure 3: Simulation of the experiments of Kovács and Julesz (1993). Stimuli consisted of 2025 randomly oriented Gabor patches, with 24 elements aligned to form a contour. Each data point represents results from 500 trials. a) Plot of the performance of the model with respect to the ratio of the separation of the background elements to the contour elements. Results show closed contours are salient to a more salient than open contours. b) Changes in salience as additional elements are added to open and closed contours. Results show that the salience of open contours increase monotonically while the salience of closed contours only change with the addition of the last element. Open contours were initially made up of 7 elements while closed contours were made up of 17 elements. c) The data from Kovács and Julesz (1993) are re-plotted for comparison.

## 3.3 REAL IMAGES

A stringent test of the model's capabilities is the ability to extract perceptually salient contours in real images. Figure 4 and 5 show results for a typical image. The original grayscale image, the output of the steerable filters, and the output of the model are shown in Figure 4a,b,c and Figure 5a,b,c respectively. The network is able to extract some of the more salient contours and ignore other high contrast edges detected by the steerable filters. Both simulations used filters at only one spatial scale and could be improved through interactions across multiple spatial frequencies. Nevertheless, the model shows promise for automated image processing applications

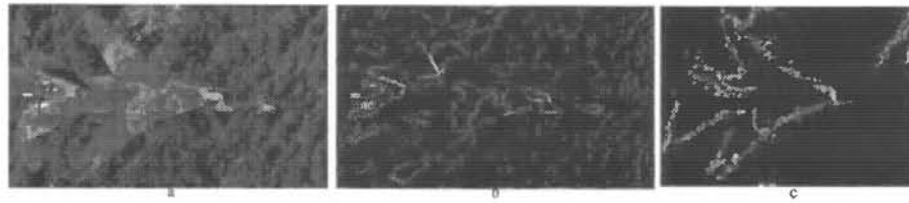

Figure 4: a) Plane image. b) Steerable filter response. c) Result of model showing the most salient contours.

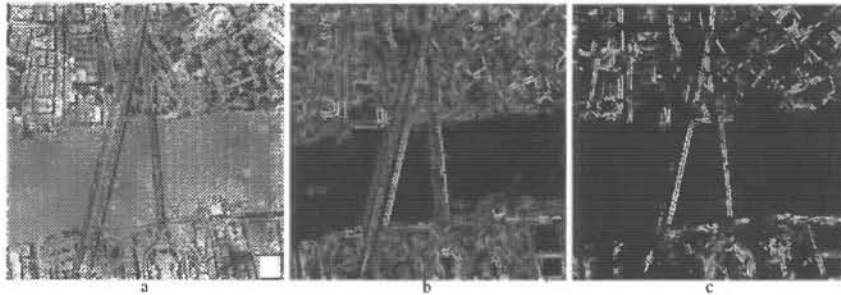

Figure 5: a) Satellite image of Bangkok. b) Steerable filter response. c) Salient contours extracted from the image. The model included filters at only one spatial frequency.

## 4    CONCLUSION

We have presented a cortically-based model that is able to identify perceptually salient contours in images containing high levels of noise. The model is based on the use of long distance intracortical connections that facilitate the responses of cells lying along smooth contours. Salience is defined as the combined activity of the synchronized population of cells responding to a particular contour. The model qualitatively accounts for a range of physiological and psychophysical results and can be used in extracting salient contours in real images.

### Acknowledgements

Supported by the Office of Naval Research (N00014-93-1-0681), The Whitaker Foundation, and the McDonnell-Pew Program in Cognitive Neuroscience.

### References

Bair, W., Koch, C., Newsome, W. & Britten, K. (1994). Power spectrum analysis of bursting cells in area MT in the behaving monkey. *Journal of Neuroscience, 14,* 2870-2892.

Baldi, P. & Meir, R. (1990). Computing with arrays of coupled oscillators: An application to preattentive texture discrimination. *Neural Computation, 2,* 458-471.

Ermentrout, G. B. (1985). The behavior of rings of coupled oscillators. *Journal of Mathematical Biology, 23,* 55-74.

Field, D. J., Hayes, A. & Hess, R. F. (1993). Contour integration by the human visual system: Evidence for a local "Association Field". *Vision Research, 33,* 173-193.

Fitzpatrick, D. (1996). The functional-organization of local circuits in visual-cortex – insights from the study of tree shrew striate cortex. *Cerebral Cortex, 6,* 329-341.

Freeman, W. T. & Adelson, E. H. (1991). The design and use of steerable filters. *IEEE Transactions on Pattern Analysis and Machine Intelligence, 13,* 891-906.

Gilbert, C. D., Das, A., Ito, M., Kapadia, M. & Westheimer, G. (1996). Spatial integration and cortical dynamics. *Proceedings of the National Academy of Sciences USA, 93*, 615-622.

Ghose, G. M. & Freeman, R. D. (1992). Oscillatory discharge in the visual system: Does it have a functional role? *Journal of Neurophysiology, 68*, 1558-1574.

Gray, C. M. & McCormick, D. A. (1996). Chattering cells -- superficial pyramidal neurons contributing to the generation of synchronous oscillations in the visual-cortex. *Science, 274*, 109-113.

Kapadia, M. K., Ito, M., Gilbert, C. D. & Westheimer. G. (1995). Improvement in visual sensitivity by changes in local context: Parallel studies in human observers and in V1 of alert monkeys. *Neuron,15*, 843-856.

Kopell, N. & Ermentrout, G. B. (1986). Symmetry and phaselocking in chains of weakly coupled oscillators. *Communications on Pure and Applied Mathematics, 39*, 623-660.

Kovács, I. & Julesz, B. (1993). A closed curve is much more than an incomplete one: Effect of closure in figure-ground segmentation. *Proceedings of National Academy of Sciences, USA, 90*, 7495-7497.

Kovács, I. & Julesz, B. (1994). Perceptual sensitivity maps within globally defined visual shapes. *Nature, 370*, 644-646.

Kovács, I., Polat, U. & Norcia, A. M. (1996). Breakdown of binding mechanisms in amblyopia. *Investigative Ophthalmology & Visual Science, 37*, 3078.

Lund, J., Fitzpatrick, D. & Humphrey, A. L. (1985). The striate visual cortex of the tree shrew. In Jones, E. G. & Peters, A. (Eds), *Cerebral Cortex* (pp. 157-205). New York: Plenum.

Nelson, J. I. & Frost, B. J. (1985). Intracortical facilitation among co-oriented, co-axially aligned simple cells in cat striate cortex. *Experimental Brain Research, 61*, 54-61.

Parent, P. & Zucker, S. W. (1989). Trace inference, curvature consistency, and curve detection. *IEEE Transactions on Pattern Analysis and Machine Intelligence, 11*, 823-839.

Polat, U. & Sagi, D. (1993). Lateral interactions between spatial channels: Suppression and facilitation revealed by lateral masking experiments. *Vision Research, 33*, 993-999.

Polat, U. & Sagi, D. (1994). The architecture of perceptual spatial interactions. *Vision Research, 34*, 73-78.

Rockland, K. S. & Lund, J. S. (1983). Intrinsic laminar lattice connections in primate visual cortex. *Journal of Comparative Neurology, 216*, 303-318.

Singer, W. & Gray, C. M. (1995). Visual feature integration and the temporal correlation hypothesis. *Annual Review of Neuroscience, 18*, 555-586.

Somers, D. & Kopell, N. (1993). Rapid synchronization through fast threshold modulation. *Biological Cybernetics, 68*, 393-407.

Wang, D. (1995). Emergent synchrony in locally coupled neural oscillators. *IEEE Transactions on Neural Networks, 6*, 941-948.

Weliky M., Kandler, K., Fitzpatrick, D. & Katz, L. C. (1995). Patterns of excitation and inhibition evoked by horizontal connections in visual cortex share a common relationship to orientation columns. *Neuron, 15*, 541-552.
